# Evolving Learnable Languages

**Bradley Tonkes**
Dept of Comp. Sci. and Elec. Engineering
University of Queensland
Queensland, 4072
Australia
*btonkes@csee.uq.edu.au*

**Alan Blair**
Department of Computer Science
University of Melbourne
Parkville, Victoria, 3052
Australia
*blair@cs.mu.oz.au*

**Janet Wiles**
Dept of Comp. Sci. and Elec. Engineering
School of Psychology
University of Queensland
Queensland, 4072
Australia
*janetw@csee.uq.edu.au*

## Abstract

Recent theories suggest that language acquisition is assisted by the evolution of languages towards forms that are easily learnable. In this paper, we evolve combinatorial languages which can be learned by a recurrent neural network quickly and from relatively few examples. Additionally, we evolve languages for generalization in different "worlds", and for generalization from specific examples. We find that languages can be evolved to facilitate different forms of impressive generalization for a minimally biased, general purpose learner. The results provide empirical support for the theory that the language itself, as well as the language environment of a learner, plays a substantial role in learning: that there is far more to language acquisition than the language acquisition device.

## 1 Introduction: Factors in language learnability

In exploring issues of language learnability, the special abilities of humans to learn complex languages have been much emphasized, with one dominant theory based on innate, domain-specific learning mechanisms specifically tuned to learning human languages. It has been argued that without strong constraints on the learning mechanism, the complex syntax of language could not be learned from the sparse data that a child observes [1]. More recent theories challenge this claim and emphasize the interaction between learner and environment [2]. In addition to these two theories is the proposal that rather than "language-savvy infants", languages themselves adapt to human learners, and the ones that survive are "infant-friendly languages" [3–5]. To date, relatively few empirical studies have explored how such adaptation of language facilitates learning. Hare and Elman [6] demonstrated that

classes of past tense forms could evolve over simulated generations in response to changes in the frequency of verbs, using neural networks. Kirby [7] showed, using a symbolic system, how compositional languages are more likely to emerge when learning is constrained to a limited set of examples. Batali [8] has evolved recurrent networks that communicate simple structured concepts.

Our argument is not that humans are general purpose learners. Rather, current research questions require exploring the nature and extent of biases that learners bring to language learning, and the ways in which languages exploit those biases [2]. Previous theories suggesting that many aspects of language were unlearnable without strong biases are gradually breaking down as new aspects of language are shown to be learnable with much weaker biases. Studies include the investigation of how languages may exploit biases as subtle as attention and memory limitations in children [9]. A complementary study has shown that general purpose learners can evolve biases in the form of initial starting weights that facilitate the learning of a family of recursive languages [10].

In this paper we present an empirical paradigm for continuing the exploration of factors that contribute to language learnability. The paradigm we propose necessitates the evolution of languages comprising recursive sentences over symbolic strings — languages whose sentences cannot be conveyed without combinatorial composition of symbols drawn from a finite alphabet. The paradigm is not based on any specific natural language, but rather, it is the simplest task we could find to illustrate the point that languages with compositional structure can be evolved to be learnable from few sentences. The simplicity of the communication task allows us to analyze the language and its generalizability, and highlight the nature of the generalization properties.

We start with the evolution of a recursive language that can be learned easily from five sentences by a minimally biased learner. We then address issues of robust learning of evolved languages, showing that different languages support generalization in different ways. We also address a factor to which scant regard has been paid, namely that languages may evolve not just to their learners, but also to be easily generalizable from a specific set of concepts. It seems almost axiomatic that learning paradigms should sample randomly from the training domain. It may be that human languages are not learnable from random sentences, but are easily generalizable from just those examples that a child is likely to be exposed to in its environment. In the third series of simulations, we test whether a language can adapt to be learnable from a core set of concepts.

## 2   A paradigm for exploring language learnability

We consider a simple language task in which two recurrent neural networks try to communicate a "concept" represented by a point in the unit interval, $[0,1]$ over a symbolic channel. An *encoder* network sends a sequence of symbols (thresholded outputs) for each concept, which a *decoder* network receives and processes back into a concept (the framework is described in greater detail in [11]). For communication to be successful, the decoder's output should approximate the encoder's input for all concepts.

The architecture for the encoder is a recurrent network with one input unit and five output units, and with recurrent connections from both the output and hidden units back to the hidden units. The encoder produces a sequence of up to five symbols (states of the output units) taken from $\Sigma = \{A, ..., J\}$, followed by the \$ symbol, for each concept taken from $[0,1]$. To encode a value $x \in [0,1]$, the network

Figure 1: Hierarchical decomposition of the language produced by an encoder, with the first symbols produced appearing near the root of the tree. The ordering of leaves in the tree represent the input space, smaller inputs being encoded by those sentences on the left. The examples used to train the best decoder found during evolution are highlighted. The decoder must generalize to all other branches. In order to learn the task, the decoder must generalize systematically to novel states in the tree, including generalizing to symbols in different positions in the sequence. (Figure 2 shows the sequence of states of a successful decoder.)

is presented with a sequence of inputs $(x, 0, 0, ..)$. At each step, the output units of the network assume one of eleven states: all zero if no output is greater than 0.5 (denoted by $); or the saturation of the two highest activations at 1.0 and the remainder at 0.0 (denoted by $A = [1, 1, 0, 0, 0]$ through $J = [0, 0, 0, 1, 1]$). If the zero output is produced, propagation is halted. Otherwise propagation continues for up to five steps, after which the output units assume the zero ($) state.

The decoder is a recurrent network with 5 input units and a single output, and a recurrent hidden layer. Former work [11] has shown that due to conflicting constraints of the encoder and decoder, it is easier for the decoder to process strings which are in the *reverse order* to those produced by the encoder. Consequently, the input to the decoder is taken to be the reverse of the output from the decoder, except for $, which remains the last symbol. (For clarity, strings are written in the order produced by the encoder.) Each input pattern presented to the decoder matches the output of the encoder — either two units are active, or none are. The network is trained with backpropagation through time to produce the desired value, $x$, on presentation of the final symbol in the sequence ($).

A simple hill-climbing evolutionary strategy with a two-stage evaluation function is used to evolve an initially random encoder into one which produces a language which a random decoder can learn easily from few examples. The evaluation of an encoder, mutated from the current "champion" by the addition of Gaussian noise to the weights, is performed against two criteria. (1) The mutated network must produce a greater variety of sequences over the range of inputs; and (2) a decoder with initially small random weights, trained on the mutated encoder's output, must yield lower sum-squared error across the entire range of inputs than the champion.

Each mutant encoder is paired with a single decoder with initially random weights. If the mutant encoder-decoder pair is more successful than the champion, the mutant becomes champion and the process is repeated. Since the encoder's input space is continuous and impossible to examine in its entirety, the input range is approximated with 100 uniformly distributed examples from 0.00 to 0.99. The final output from the hill-climber is the language generated by the best encoder found.

## 2.1   Evolving an easily learnable language

Humans learn from sparse data. In the first series of simulations we test whether a compositional language can be evolved that learners can reliably and effectively learn from only five examples. From just five training examples, it seems unreasonable to expect that any decoder would learn the task. The task is intentionally hard in that a language is restricted to sequences of discrete symbols with which it must describe a continuous space. Note that simple linear interpolation is not possible due to the symbolic alphabet of the languages. Recursive solutions are possible but are unable to be learned by an *unbiased* learner. The decoder is a *minimally-biased* learner and as the simulations showed, performed much better than arguments based on learnability theory would predict.

Ten languages were evolved with the hill-climbing algorithm (outlined above) for 10000 generations.[1] For each language, 100 new random decoders were trained under the same conditions as during evolution (five examples, 400 epochs). All ten runs used encoders and decoders with five hidden units.

All of the evolved languages were learnable by some decoders (minimum 20, maximum 72, mean 48). A learner is said to have effectively learned the language if its sum-squared-error across the 100 points in the space is less than 1.0.[2] Encoders employed on average 36 sentences (minimum 21, maximum 60) to communicate the 100 points. The 5 training examples for each decoder were sampled randomly from $[0, 1]$ and hence some decoders faced very difficult generalization tasks. The difficulty of the task is demonstrated by the language analyzed in Figures 1 and 2. The evolved languages all contained similar compositional structure to that of the language described in Figures 1 and 2. The inherent biases of the decoder, although minimal, are clearly sufficient for learning the compositional structure.

## 3   Evolving languages for particular generalization

The first series of simulations demonstrate that we can find languages for which a minimally biased learner can generalize from few examples. In the next simulations we consider whether languages can be evolved to facilitate specific forms of generalization in their users. Section 2.1 considered the case where the decoder's required output was the same as the encoder's input. This setup yields the approximation to the line $y = x$ in Figure 2. The compositional structure of the evolved languages allows the decoder to generalize to unseen regions of the space. In the following series of simulations we consider the relationship between the structure of a language and the way in which the decoder is required to generalize. This association is studied by altering the desired relationship between the encoder's input $(x)$ and the decoder's output $(y)$.

Two sets of ten languages were evolved, one set requiring $y = x$ (*identity*, as in section 2.1), the other using a function resembling a series of five steps at random heights: $y = r(\lfloor 5x \rfloor); r = (0.3746, 0.5753, 0.8102, 0.7272, 0.4527)$ (*random step*)[3]. All conditions were as for section 2, with the exception that 10 training examples were used and the hill-climber ran for 1000 generations. On completion of evolution, 100 decoders were trained on the 20 final languages under both conditions above as

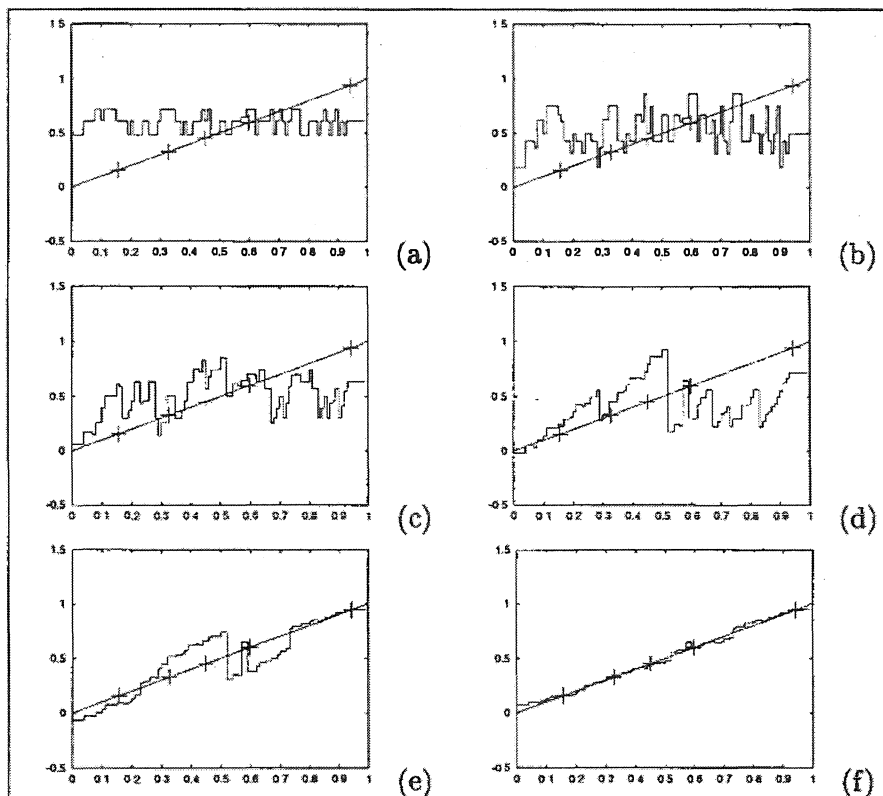

Figure 2: Decoder output after seeing the first $n$ symbols in the message, for $n = 1$ (a) to $n = 6$ (f) (from the language in Figure 1). The X-axis is the encoder's input, the Y-axis is the decoder's output at that point in the sequence. The five points that the decoder was trained on are shown as crosses in each graph. After the first symbol ($A$, $B$, $C$, $E$ or $\$$), the decoder outputs one of five values (a); after the second symbol, more outputs are possible (b). Subsequent symbols in each string specify finer gradations in the output. Note that the output is not constructed monotonically, with each symbol providing a closer approximation to the target function, but rather recursively, only approximating the linear target at the final position in each sequence. Structure inherent in the sequences allows the system to generalize to parts of the space it has never seen. Note that the generalization is not based on interpolation between symbol values, but rather on their compositional structure.

well as two others, a *sine* function and a *cubic* function.

The results show that languages can be evolved to enhance generalization preferentially for one "world" over another. On average, the languages performed far better when tested in the world in which they were evolved than in other worlds. Languages evolved for the identity mapping were on average learned by 64% of decoders trained on the identity task compared with just 5% in the random step case. Languages evolved for the random step task were learned by 60% of decoders trained on the random step task but only 24% when trained on the identity task. Decoders generally performed poorly on the cubic function, and no decoder learned the sine task from either set of evolved languages. The second series of simulations show that the manner in which the decoder generalizes is not restricted to the task of section 2.1. Rather, the languages evolve to facilitate generalization by the decoder in different ways, aided by its minimal biases.

# 4  Generalization from core concepts

In the former simulations, randomly selected concepts were used to train decoders. In some cases a pathological distribution of points made learning extremely difficult. In contrast, it seems likely that human children learn language based on a common set of semantically-constrained core concepts ("Mom", "I want milk", "no", etc). For the third series of simulations, we tested whether selecting a fortuitous set of training concepts could have a positive affect on the success of an evolved language. The simulations with alternative generalization functions (section 3) indicated that decoders had difficulty generalizing to the sine function. Even when encoders were evolved specifically on the sine task, in the best of 10 systems only 13 of 100 random decoders successfully learned.

We evolved a new language on a specifically chosen set of 10 points for generalization to the sine function. One hundred decoders were then trained on the resulting language using either the same set of 10 points, or a random set. Of the networks trained on the fixed set, 92 learned the tasked, compared with 5 networks trained on the random sets. That a language evolves to communicate a restricted set of concepts is not particularly unusual. But what this simulation shows is the more surprising result that a language can evolve to generalize from specific core concepts to a whole recursive language in a particular way (in this case, a sine function).

# 5  Discussion

The first series of simulations show that a compositional language can be learned from five strings by an recurrent network. Generalization performance included correct decoding of novel branches and symbols in novel positions (Figure 1). The second series of simulations highlight how a language can be evolved to facilitate different forms of generalization in the decoder. The final simulation demonstrates that languages can also be tailored to generalize from a specific set of examples.

The three series of simulations modify the language environment of the decoder in three different ways: (1) the relationship between utterances and meaning; (2) the type of generalization required from the decoder; and (3) the particular utterances and meanings to which a learner is exposed. In each case, the language environment of the learner was sculpted to exploit the minimal biases present in the learner. While taking an approach similar to [10] of giving the learner an additional bias in the form of initial weights was also likely to have been effective, the purpose of the simulations was to investigate how strongly external factors could assist in simplifying learning.

# 6  Conclusions

> "The key to understanding language learnability does not lie in the richly social context of language training, nor in the incredibly prescient guesses of young language learners; rather, it lies in a process that seems otherwise far remote from the microcosm of toddlers and caretakers — language change. Although the rate of social evolutionary change in learning structure appears unchanging compared to the time it takes a child to develop language a-bilities, this process is crucial to understanding how the child can learn a language that on the surface appears impossibly complex and poorly taught." [3, p115].

In this paper we studied ways in which languages can adapt to their learners. By running simulations of a language evolution process, we contribute additional components to the list of aspects of language that can be learned by minimally-biased, general-purpose learners, namely that recursive structure can be learned from few examples, that languages can evolve to facilitate generalization in a particular way, and that they can evolve to be easily learnable from common sentences. In all the simulations in this paper, enhancement of language learnability is achieved through changes to the learner's environment without resorting to adding biases in the language acquisition device.

## Acknowledgements

This work was supported by an APA to Bradley Tonkes, a UQ Postdoctoral Fellowship to Alan Blair and an ARC grant to Janet Wiles.

## Footnotes

[1]One generation represents the creation of a more variable, mutated encoder and the subsequent training of a decoder.

[2]A language is said to be reliably learnable when at least 50% of random decoders are able to effectively learn it within 400 epochs.

[3]$\lfloor 5x \rfloor$ provides an index into the array $r$, based on the magnitude of $x$.

## References

[1] N. Chomsky. *Language and Mind.* Harcourt, Brace, New York, 1968.

[2] J. L. Elman, E. A. Bates, M. H. Johnson, A. Karmiloff-Smith, D. Parisi, and K. Plunkett. *Rethinking Innateness: A Connectionist Perspective on Development.* MIT Press, Boston, 1996.

[3] T. W. Deacon. *The Symbolic Species: The Co-Evolution of Language and the Brain.* W. W. Norton and Company, New York, 1997.

[4] S. Kirby. Fitness and the selective adaptation of language. In J. Hurford, C. Knight, and M. Studdert-Kennedy, editors, *Approaches to the Evolution of Language.* Cambridge University Press, Cambridge, 1998.

[5] M. H. Christiansen. Language as an organism — implications for the evolution and acquisition of language. Unpublished manuscript, February 1995.

[6] M. Hare and J. L. Elman. Learning and morphological change. *Cognition*, 56:61–98, 1995.

[7] S. Kirby. Syntax without natural selection: How compositionality emerges from vocabulary in a population of learners. In C. Knight, J. Hurford, and M. Studdert-Kennedy, editors, *The Evolutionary Emergence of Language: Social function and the origins of linguistic form.* Cambridge University Press, Cambridge, 1999.

[8] J. Batali. Computational simulations of the emergence of grammar. In J. Hurford, C. Knight, and M. Studdert-Kennedy, editors, *Approaches to the Evolution of Language*, pages 405–426. Cambridge University Press, Cambridge, 1998.

[9] J. L. Elman. Learning and development in neural networks: The importance of starting small. *Cognition*, 48:71–99, 1993.

[10] J. Batali. Innate biases and critical periods: Combining evolution and learning in the acquisition of syntax. In R. Brooks and P. Maes, editors, *Proceedings of the Fourth Artificial Life Workshop*, pages 160–171. MIT Press, 1994.

[11] B. Tonkes, A. Blair, and J. Wiles. A paradox of neural encoders and decoders, or, why don't we talk backwards? In B. McKay, X. Yao, C. S. Newton, J. -H. Kim, and T. Furuhashi, editors, *Simulated Evolution and Learning*, volume 1585 of *Lecture Notes in Artificial Intelligence.* Springer, 1999.

